# SPALS: Fast Alternating Least Squares via Implicit Leverage Scores Sampling

**Dehua Cheng**
University of Southern California
dehua.cheng@usc.edu

**Richard Peng**
Georgia Institute of Technology
rpeng@cc.gatech.edu

**Ioakeim Perros**
Georgia Institute of Technology
perros@gatech.edu

**Yan Liu**
University of Southern California
yanliu.cs@usc.edu

## Abstract

Tensor CANDECOMP/PARAFAC (CP) decomposition is a powerful but computationally challenging tool in modern data analytics. In this paper, we show ways of sampling intermediate steps of alternating minimization algorithms for computing low rank tensor CP decompositions, leading to the sparse alternating least squares (SPALS) method. Specifically, we sample the Khatri-Rao product, which arises as an intermediate object during the iterations of alternating least squares. This product captures the interactions between different tensor modes, and form the main computational bottleneck for solving many tensor related tasks. By exploiting the spectral structures of the matrix Khatri-Rao product, we provide efficient access to its statistical leverage scores. When applied to the tensor CP decomposition, our method leads to the first algorithm that runs in sublinear time per-iteration and approximates the output of deterministic alternating least squares algorithms. Empirical evaluations of this approach show significant speedups over existing randomized and deterministic routines for performing CP decomposition. On a tensor of the size $2.4m \times 6.6m \times 92k$ with over 2 billion nonzeros formed by Amazon product reviews, our routine converges in two minutes to the same error as deterministic ALS.

## 1 Introduction

Tensors, a.k.a. multidimensional arrays, appear frequently in many applications, including spatial-temporal data modeling [40], signal processing [12, 14], deep learning [29] and more. Low-rank tensor decomposition [21] is a fundamental tool for understanding and extracting the information from tensor data, which has been actively studied in recent years. Developing scalable and provable algorithms for most tensor processing tasks is challenging due to the non-convexity of the objective [18, 21, 16, 1]. Especially in the era of big data, scalable low-rank tensor decomposition algorithm (that runs in nearly linear or even sublinear time in the input data size) has become an absolute must to command the full power of tensor analytics. For instance, the Amazon review data [24] yield a $2,440,972 \times 6,643,571 \times 92,626$ tensor with 2 billion nonzero entries after preprocessing. Such data sets pose challenges of scalability to some of the simplest tensor decomposition tasks.

There are multiple well-defined tensor ranks[21]. In this paper, we focus on the tensor CANDECOMP/PARAFAC (CP) decomposition [17, 3], where the low-rank tensor is modeled by the summation over many rank-1 tensors. Due to its simplicity and interpretability, tensor CP decomposition, which is to find the best rank-$R$ approximation for the input tensor often by minimizing the square loss function, has been widely adopted in many applications [21].

Matrix Khatri-Rao (KRP) product captures the interactions between different tensor modes in the CP decomposition, and it is essential for understanding many tensor related tasks. For instance, in the alternating least square (ALS) algorithm, which has been the workhorse for solving the tensor CP decomposition problem, a compact representation of the KRP can reduce the computational cost directly. ALS is a simple and parameter-free algorithm that optimizes the target rank-$R$ tensor by updating its factor matrices in the block coordinate descent fashion. In each iteration, the computational bottleneck is to solve a least square regression problem, where the size of the design matrix, a KRP of factor matrices, is $n^2 \times n$ for an $n \times n \times n$ tensor. While least square regression is one of the most studied problem, solving it exactly requires at least $\mathcal{O}(n^2)$ operations [23], which can be larger than the size of input data for sparse tensors. For instance, the amazon review data with $2 \times 10^9$ nonzeros leads to a computational cost on the order of $10^{12}$ per iteration. Exploiting the structure of the KRP can reduce this cost to be linear in the input size, which on large-scale applications is still expensive for an iterative algorithm.

An effective way for speeding up such numerical computations is through randomization [23, 38], where the computational cost can be uncorrelated with the ambient size of the input data in the optimal case. By exploring the connection between the spectral structures of the design matrix as the KRP of the factor matrices, we provide efficient access to the *statistical leverage score* of the design matrix. It allows us to propose the SPALS algorithm that samples rows of the KRP in a nearly-optimal manner. This near optimality is twofold: 1) the estimates of leverage scores that we use have many tight cases; 2) the operation of sampling a row can be efficiently performed. The latter requirement is far from trivial: Note that even when the optimal sampling probability is given, drawing a sample may require $\mathcal{O}(n^2)$ preprocessing. Our result on the spectral structures of the design matrix allows us to achieve both criteria simultaneously, leading to the first sublinear-per-iteration cost ALS algorithm with provable approximation guarantees. Our contributions can be summarized as follows:

1. We show a close connection between the statistical leverage scores of the matrix Khatri-Rao product and the scores of the input matrices. This yields efficient and accurate leverage score estimations for importance sampling;

2. Our algorithm achieves the state-of-art computational efficiency, while approximating the ALS algorithm provably for computing CP tensor decompositions. The running time of each iteration of our algorithm is $\tilde{O}(nR^3)$, *sublinear* in the input size for large tensors.

3. Our theoretical results on the spectral structure of KRP can also be applied on other tensor related applications such as stochastic gradient descent [26] and high-order singular value decompositions (HOSVD) [13].

We formalize the definitions in Section 2 and present our main results on leverage score estimation of the KRP in Section 3. The SPALS algorithm and its theoretical analysis are presented in Section 4. We discuss connections with previous works in Section 5. In Section 6, we empirical evaluate this algorithm and its variants on both synthetic and real world data. And we conclude and discuss our work in Section 7.

## 2   Notation and Background

Vectors are represented by boldface lowercase letters, such as, $\mathbf{a}, \mathbf{b}, \mathbf{c}$; Matrices are represented by boldface capital letters, such as, $\mathbf{A}, \mathbf{B}, \mathbf{C}$; Tensors are represented by boldface calligraphic capital letters, such as, $\boldsymbol{\mathcal{T}}$. Without loss of generality, in this paper we focus our discussion for the 3-mode tensors, but our results and algorithm can be easily generalized to higher-order tensors.

The $i$th entry of a vector is denoted by $a_i$, element $(i, j)$ of a matrix $\mathbf{A}$ is denoted by $A_{ij}$, and the element $(i, j, k)$ of a tensor $\boldsymbol{\mathcal{T}} \in \mathbb{R}^{I \times J \times K}$ is denoted by $\mathcal{T}_{ijk}$. For notation simplicity, we assume that $(i, j)$ also represents the index $i + Ij$ between 1 and $IJ$, where the value $I$ and $J$ should be clear from the context.

For a tensor $\boldsymbol{\mathcal{T}} \in \mathbb{R}^{I \times J \times K}$, we denote the tensor norm as $\|\boldsymbol{\mathcal{T}}\|$, i.e., $\|\boldsymbol{\mathcal{T}}\| = \sqrt{\sum_{i,j,k=1}^{I,J,K} \mathcal{T}_{ijk}^2}$.

**Special Matrix Products** Our manipulation of tensors as matrices revolves around several matrix products. Our main focus is the matrix *Khatri-Rao product* (KRP) $\odot$, where for a pair of matrices $\mathbf{A} \in \mathbb{R}^{I \times R}$ and $\mathbf{B} \in \mathbb{R}^{J \times R}$, $\mathbf{A} \odot \mathbf{B} \in \mathbb{R}^{(IJ) \times R}$ has element $((i, j), r)$ as $\mathbf{A}_{ir}\mathbf{B}_{jr}$.

We also utilize the matrix *Kronecker product* $\otimes$ and the elementwise matrix product $*$. More details on these products can be found in Appendix A and [21].

**Tensor Matricization** Here we consider only the case of mode-$n$ matricization. For $n = 1, 2, 3$, the mode-$n$ matricization of a tensor $\mathcal{T} \in \mathbb{R}^{I \times J \times K}$ is denoted by $\mathbf{T}_{(n)}$. For instance, $\mathbf{T}_{(3)} \in \mathbb{R}^{K \times IJ}$, where the element $(k, (i, j))$ is $\mathcal{T}_{ijk}$.

**Tensor CP Decomposition** The tensor CP decomposition [17, 3] expresses a tensor as the sum of a number of rank-one tensors, e.g.,

$$\mathcal{T} = \sum_{r=1}^{R} \mathbf{a}_r \circ \mathbf{b}_r \circ \mathbf{c}_r,$$

where $\circ$ denotes the outer product, $\mathcal{T} \in \mathbb{R}^{I \times J \times K}$ and $\mathbf{a}_r \in \mathbb{R}^I, \mathbf{b}_r \in \mathbb{R}^J$, and $\mathbf{c}_r \in \mathbb{R}^K$ for $r = 1, 2, \ldots, R$. Tensor CP decomposition will be compactly represented using $[\![\mathbf{A}, \mathbf{B}, \mathbf{C}]\!]$, where the *factor matrices* $\mathbf{A} \in \mathbb{R}^{I \times R}, \mathbf{B} \in \mathbb{R}^{J \times R}$ and $\mathbf{C} \in \mathbb{R}^{K \times R}$ and $\mathbf{a}_r, \mathbf{b}_r, \mathbf{c}_r$ are their $r$-th column respectively, i.e., $[\![\mathbf{A}, \mathbf{B}, \mathbf{C}]\!]_{ijk} = \sum_{r=1}^{R} A_{ir} B_{jr} C_{kr}$. Similar as in the matrix case, each rank-1 component is usually interpreted as a hidden factor, which captures the interactions between all dimensions in the simplest way.

Given a tensor $\mathcal{T} \in \mathbb{R}^{I \times J \times K}$ along with target rank $R$, the goal is to find a rank-$R$ tensor specified by its factor matrices $\mathbf{A} \in \mathbb{R}^{I \times R}, \mathbf{B} \in \mathbb{R}^{J \times R}, \mathbf{C} \in \mathbb{R}^{K \times R}$, that is as close to $\mathcal{T}$ as possible:

$$\min_{\mathbf{A}, \mathbf{B}, \mathbf{C}} \|\mathcal{T} - [\![\mathbf{A}, \mathbf{B}, \mathbf{C}]\!]\|^2 = \sum_{i, j, k} \left( \mathcal{T}_{i, j, k} - \sum_{r=1}^{R} A_{ir} B_{jr} C_{kr} \right)^2.$$

**Alternating Least Squares Algorithm** A widely used method for performing CP decomposition is alternating least squares (ALS) algorithm. It iteratively minimizes one of the factor matrices with the others fixed. For instance, when the factors $\mathbf{A}$ and $\mathbf{B}$ are fixed, algebraic manipulations suggest that the best choice of $\mathbf{C}$ can be obtained by solving the least squares regression:

$$\min_{\mathbf{C}} \left\| \mathbf{X}\mathbf{C}^\top - \mathbf{T}_{(3)}^\top \right\|^2, \tag{1}$$

where the design matrix $\mathbf{X} = \mathbf{B} \odot \mathbf{A}$ is the KRP of $\mathbf{A}$ and $\mathbf{B}$, and $\mathbf{T}_{(3)}$ is the matricization of $\mathcal{T}$ [21].

## 3 Near-optimal Leverage Score Estimation for Khatri-Rao Product

As shown in Section 2, the matrix KRP captures the essential interactions between the factor matrices in the tensor CP decomposition. This task is challenging because the size of KRP of two matrices is significantly larger than the input matrices. For example, for the amazon review data, the KRP of two factor matrices contains $10^{12}$ rows, which is much larger than the data set itself with $10^9$ nonzeros.

Importance sampling is one of the most powerful tools for obtaining sample efficient randomized data reductions with strong guarantees. However, effective implementation requires comprehensive knowledge on the objects to be sampled: the KRP of factor matrices. In this section, we provide an efficient and effective toolset for estimating the *statistical leverage scores* of the KRP of factor matrices, giving a direct way of applying importance sampling, one of the most important tools in randomized matrix algorithms, for tensor CP decomposition related applications.

In the remainder of this section, we first define and discuss the optimal importance: *statistical leverage score*, in the context of $\ell_2$-regression. Then we propose and prove our near-optimal leverage score estimation routine.

### 3.1 Leverage Score Sampling for $\ell_2$-regression

It is known that, when $p \ll n$, subsampling the rows of design matrix $\mathbf{X} \in \mathbb{R}^{n \times p}$ by its statistical leverage score and solving on the samples provides efficient approximate solution to the least square regression problem: $\min_{\beta} \|\mathbf{X}\beta - \mathbf{y}\|_2^2$, with strong theoretical guarantees [23].

**Definition 3.1** (Statistical Leverage Score). *Given an $n \times r$ matrix $\mathbf{X}$, with $n > r$, let $\mathbf{U}$ denote the $n \times r$ matrix consisting of the top-$r$ left singular vectors of $\mathbf{X}$. Then, the quantity*

$$\tau_i = \|\mathbf{U}_{i,:}\|_2^2,$$

*where $\mathbf{U}_{i,:}$ denotes the $i$-th row of $\mathbf{U}$, is the* statistical leverage score *of the $i$-th row of $\mathbf{X}$.*

The statistical leverage score of a certain row captures importance of the row in forming the linear subspace. Its optimality in solving $\ell_2$-regression can be explained by the subspace projection nature of linear regression.

It does not yield an efficient algorithm for the optimization problem in Equation (1) due to the difficulties of computing *statistical leverage scores*. But this reduction to the matrix setting allows for speedups using a variety of tools. In particular, sketching [6, 25, 27] or iterative sampling [22, 9] lead to routines that run in input sparsity time: $O(nnz)$ plus the cost of solving an $O(r \log n)$ sized least squares problem. However, directly applying these methods still require at least one pass over $\mathcal{T}$ at each iteration, which will dominate the overall cost.

## 3.2 Near-optimal Leverage Score Estimation

As discussed in the previous section, the KRPs of factor matrices capture the interaction between two modes in the tensor CP decomposition, e.g., the design matrix $\mathbf{B} \odot \mathbf{A}$ in the linear regression problem. To extract a compact representation of the interaction, the statistical leverage scores of $\mathbf{B} \odot \mathbf{A}$ provide an informative distribution over the rows, which can be utilized to select the important subsets of rows randomly.

For a matrix with $IJ$ rows in total, e.g., $\mathbf{B} \odot \mathbf{A}$, in general, the calculation of statistical leverage score is prohibitively expensive. However, due to the special structure of the KRP $\mathbf{B} \odot \mathbf{A}$, the upper bound of statistical leverage score, which is sufficient to obtain the same guarantee by using slightly more samples, can be efficiently estimated, as shown in Theorem 3.2.

**Theorem 3.2** (Khatri-Rao Bound). *For matrix $\mathbf{A} \in \mathbb{R}^{I \times R}$ and matrix $\mathbf{B} \in \mathbb{R}^{J \times R}$, where $I > R$ and $J > R$, let $\tau_i^{\mathbf{A}}$ and $\tau_j^{\mathbf{B}}$ be the statistical leverage score of the $i$-th and $j$-th row of $\mathbf{A}$ and $\mathbf{B}$, respectively. Then, for statistical leverage score of the $(iJ + j)$-th row of matrix $\mathbf{A} \odot \mathbf{B}$, $\tau_{i,j}^{\mathbf{A} \odot \mathbf{B}}$, we have*

$$\tau_{i,j}^{\mathbf{A} \otimes \mathbf{B}} \leq \tau_i^{\mathbf{A}} \tau_j^{\mathbf{B}}.$$

*Proof.* Let the singular value decomposition of $\mathbf{A}$ and $\mathbf{B}$ be $\mathbf{A} = \mathbf{U}^a \mathbf{\Lambda}^a \mathbf{V}^{a\top}$ and $\mathbf{B} = \mathbf{U}^b \mathbf{\Lambda}^b \mathbf{V}^{b\top}$, where $\mathbf{U}^a \in \mathbb{R}^{I \times R}$, $\mathbf{U}^b \in \mathbb{R}^{J \times R}$, and $\mathbf{\Lambda}^a, \mathbf{\Lambda}^b, \mathbf{V}^a, \mathbf{V}^b \in \mathbb{R}^{R \times R}$.

By the definition of Khatri-Rao product, we have that

$$\mathbf{A} \odot \mathbf{B} = [\mathbf{A}_{:,1} \otimes \mathbf{B}_{:,1}, \ldots, \mathbf{A}_{:,R} \otimes \mathbf{B}_{:,R}] \in \mathbb{R}^{IJ \times R},$$

where $\otimes$ is the Kronecker product. By the form of SVD and Lemma B.1, we have

$$\mathbf{A} \odot \mathbf{B} = [\mathbf{U}^a \mathbf{\Lambda}^a (\mathbf{V}_{1,:}^a)^\top \otimes \mathbf{U}^b \mathbf{\Lambda}^b (\mathbf{V}_{1,:}^b)^\top, \ldots, \mathbf{U}^a \mathbf{\Lambda}^a (\mathbf{V}_{R,:}^a)^\top \otimes \mathbf{U}^b \mathbf{\Lambda}^b (\mathbf{V}_{R,:}^b)^\top]$$

$$= \left[ (\mathbf{U}^a \mathbf{\Lambda}^a) \otimes (\mathbf{U}^b \mathbf{\Lambda}^b) \right] \left( \mathbf{V}^{a\top} \odot \mathbf{V}^{b\top} \right) = \left[ \mathbf{U}^a \otimes \mathbf{U}^b \right] \left[ \mathbf{\Lambda}^a \otimes \mathbf{\Lambda}^b \right] \left( \mathbf{V}^{a\top} \odot \mathbf{V}^{b\top} \right) = \left[ \mathbf{U}^a \otimes \mathbf{U}^b \right] \mathbf{S},$$

where $\mathbf{S} = \left[ \mathbf{\Lambda}^a \otimes \mathbf{\Lambda}^b \right] \left( \mathbf{V}^{a\top} \odot \mathbf{V}^{b\top} \right) \in \mathbb{R}^{R^2 \times R}$. So the SVD of $\mathbf{A} \odot \mathbf{B}$ can be constructed using the SVD of $\mathbf{S} = \mathbf{U}_s \mathbf{\Lambda}_s \mathbf{V}_s^\top$. So the leverage score of $\mathbf{A} \odot \mathbf{B}$ can be computed from $[\mathbf{U}_a \otimes \mathbf{U}_b] \mathbf{U}_s$:

$$\mathbf{H} = [\mathbf{U}_a \otimes \mathbf{U}_b] \mathbf{U}_s \mathbf{U}_s^\top [\mathbf{U}_a \otimes \mathbf{U}_b]^\top, \tag{2}$$

and for the index $k = iJ + j$, we have

$$\tau_{i,j}^{\mathbf{A} \odot \mathbf{B}} = \mathbf{H}_{k,k} = \mathbf{e}_k^\top \mathbf{H} \mathbf{e}_k \leq \left\| \left[ \mathbf{U}^a \otimes \mathbf{U}^b \right]^\top \mathbf{e}_k \right\|_2^2 \tag{3}$$

$$= \sum_{p=1}^{R} \sum_{q=1}^{R} (\mathbf{U}_{i,p}^a)^2 (\mathbf{U}_{j,q}^b)^2 = (\sum_{p=1}^{R} (\mathbf{U}_{i,p}^a)^2)(\sum_{q=1}^{R} (\mathbf{U}_{j,q}^b)^2) = \tau_i^{\mathbf{A}} \tau_j^{\mathbf{B}}, \tag{4}$$

where $\mathbf{e}_i$ is the $i$-th natural basis vector. The first inequality is because $\mathbf{H} \preccurlyeq [\mathbf{U}_a \otimes \mathbf{U}_b] [\mathbf{U}_a \otimes \mathbf{U}_b]^\top$. $\square$

---

**Algorithm 1** Sample a row from $\mathbf{B} \odot \mathbf{A}$ and $\mathbf{T}_{(3)}$.

---

    Draw a Bernoulli random variable $z \sim \text{Bernoulli}(\beta)$.
    **if** $z = 0$ **then**
        Draw $i \sim \text{Multi}(\tau_1^{\mathbf{A}}/R, \ldots, \tau_I^{\mathbf{A}}/R)$ and $j \sim \text{Multi}(\tau_1^{\mathbf{B}}/R, \ldots, \tau_J^{\mathbf{B}}/R)$.
    **else**
        Draw a entry $(i, j, k)$ from the nonzero entries with probability proportional to $\mathcal{T}_{i,j,k}^2$.
    **end if**
    **Return** the $(jI + i)$-th row of $\mathbf{B} \odot \mathbf{A}$ and $\mathbf{T}_{(3)}$ with weight $IJp_{i,j}$.

---

For the rank-$R$ CP decomposition, the sum of the leverage scores for all rows in $\mathbf{B} \odot \mathbf{A}$ equals $R$. The sum of our upper bound relaxes it to $R^2$, which means that now we need $\tilde{O}(R^2)$ samples instead of $\tilde{O}(R)$. This result directly generalizes to the Khatri-Rao product of $k$-dimensional tensors. The proof is provided in Appendix C.

**Theorem 3.3.** *For matrices $\mathbf{A}^{(k)} \in \mathbb{R}^{I_k \times R}$ where $I_k > R$ for $k = 1, \ldots, K$, let $\tau_i^{(k)}$ be the statistical leverage score of the $i$-th row of $\mathbf{A}^{(k)}$. Then, for the $\prod_k I_k$-by-$R$ matrix $\mathbf{A}^{(1)} \odot \mathbf{A}^{(2)} \odot \cdots \odot \mathbf{A}^{(K)}$ with statistical leverage score $\tau_{i_1,\ldots,i_K}$ for the row corresponding to $\tau_{i_1,\ldots,i_K}$, we have*

$$\tau_{i_1,\ldots,i_K}^{1:K} \leq \prod_{k=1}^{K} \tau_{i_k}^{(k)},$$

*where $\tau_{i_1,\ldots,i_K}^{1:K}$ denotes the statistical leverage score of the row of $\mathbf{A}^{(1)} \odot \mathbf{A}^{(2)} \odot \cdots \odot \mathbf{A}^{(K)}$ corresponding to the $i_k$-th row of $\mathbf{A}^{(k)}$ for $k = 1, \ldots, K$.*

Our estimation enables the development of efficient numerical algorithms and is nearly optimal in three ways:

1. The estimation can be calculated in sublinear time given that $\max\{I, J, K\} = o\left(nnz(\mathcal{T})\right)$. For instance, for the amazon review data, we have $\max\{I, J, K\} \approx 10^6 \ll nnz(\mathcal{T}) \approx 10^9$;

2. The form of the estimation allows efficient sample-drawing. In fact, the row index can be drawn efficiently by considering each mode independently;

3. The estimation is tight up to a constant factor $R$. And $R$ is considered as modest constant for low-rank decomposition. Therefore, the estimation allows sample-efficient importance sampling.

## 4 SPALS: Sampling Alternating Least Squares

The direct application of our results on KRP leverage score estimation is an efficient version of the ALS algorithm for tensor CP decomposition, where the computational bottleneck is to solve the optimization problem 1.

Our main algorithmic result is a way to obtain a high quality $O(r^2 \log n)$ row sample of $\mathbf{X}$ without explicitly constructing the matrix $\mathbf{X}$. This is motivated by a recent work that implicitly generates sparsifiers for multistep random walks [4]. In particular, we sample the rows of $\mathbf{X}$, the KRP of $\mathbf{A}$ and $\mathbf{B}$, using products of quantities computed on the corresponding rows in $\mathbf{A}$ and $\mathbf{B}$, which provides a rank-1 approximation to the optimal importance: the statistical leverage scores. This leads to a sublinear time sampling routine, and implies that we can approximate the progress of each ALS step linear in the size of the factor being updated, which can be sublinear in the number of non-zeros in $\mathcal{T}$.

In the remainder of this section, we present our algorithm SPALS and prove its approximation guarantee. We will also discuss its extension to other tensor related applications.

### 4.1 Sampling Alternating Least Squares

The optimal solution to optimization problem (1) is

$$\mathbf{C} = \mathbf{T}_{(3)} \left(\mathbf{B} \odot \mathbf{A}\right) \left[\left(\mathbf{A}^{\top}\mathbf{A}\right) * \left(\mathbf{B}^{\top}\mathbf{B}\right)\right]^{-1}.$$

We separate the calculation into two parts: (1) $\mathbf{T}_{(3)}\left(\mathbf{B} \odot \mathbf{A}\right)$, and (2) $\left[\left(\mathbf{A}^{\top}\mathbf{A}\right) * \left(\mathbf{B}^{\top}\mathbf{B}\right)\right]^{-1}$, where $*$ denotes the elementwise matrix product. The latter is to invert the gram matrix of the Khatri-Rao

product, which can also be efficiently computed due to its $R \times R$ size. We will mostly focus on evaluating the former expression.

We perform the matrix multiplication by drawing a few rows from both $\mathbf{T}_{(3)}^{\top}$ and $\mathbf{B} \odot \mathbf{A}$ and construct the final solution from the subset of rows. The row of $\mathbf{B} \odot \mathbf{A}$ can be indexed by $(i, j)$ for $i = 1, \ldots, I$ and $j = 1, \ldots, J$, which correspond to the $i$-th and $j$-th row in $\mathbf{A}$ and $\mathbf{B}$, respectively. That is, our sampling problem can be seen as to sample the entries of a $I \times J$ matrix $\mathbf{P} = \{p_{i,j}\}_{i,j}$.

We define the sampling probability $p_{i,j}$ as follows,

$$p_{i,j} = (1 - \beta) \frac{\tau_i^{\mathbf{A}} \tau_j^{\mathbf{B}}}{R^2} + \beta \frac{\sum_{k=1}^{K} \boldsymbol{\mathcal{T}}_{i,j,k}^2}{\|\boldsymbol{\mathcal{T}}\|^2}. \tag{5}$$

where $\beta \in (0, 1)$. The first term is a rank-1 component for matrix $\mathbf{P}$. And when the input tensor is sparse, the second term is sparse, thus admitting the sparse plus low rank structure, which can be easily sampled as the mixture of two simple distributions. The sampling algorithm is described in Algorithm 1. Note that sampling by the leverage scores of the design matrix $\mathbf{B} \odot \mathbf{A}$ alone provides a guaranteed but worse approximation for each step [23]. Since that the design matrix itself is formed by two factor matrices, i.e., we are not directly utilizing the information in the data, we design the second term for the worst case scenario.

When $R \ll n$ and $n \ll nnz(\boldsymbol{\mathcal{T}})$, where $n = \max(I, J, K)$, we can afford to calculate $\tau_i^{\mathbf{A}}$ and $\tau_j^{\mathbf{B}}$ exactly in each iteration. So the distribution corresponding to the first term can be efficiently sampled with preparation cost $\tilde{O}(r^2 n + r^3)$ and per-sample-cost $O(\log n)$. Note that the second term requires a one-time $\mathcal{O}(nnz(\boldsymbol{\mathcal{T}}))$ preprocessing before the first iteration.

## 4.2 Approximation Guarantees

We define the following conditions:

**C1**. The sampling probability $p_{i,j}$ satisfies $p_{i,j} \geq \beta_1 \frac{\tau_{i,j}^{\mathbf{A} \odot \mathbf{B}}}{R}$ for some constant $\beta_1$;

**C2**. The sampling probability $p_{i,j}$ satisfies $p_{i,j} \geq \beta_2 \frac{\sum_{k=1}^{K} \boldsymbol{\mathcal{T}}_{i,j,k}^2}{\|\boldsymbol{\mathcal{T}}\|^2}$ for some constant $\beta_2$;

The proposed probabilities $p_{i,j}$ in Equation (5) satisfy both conditions with $\beta_1 = (1 - \beta)/R$ and $\beta_2 = \beta$. We can now prove our main approximation result.

**Theorem 4.1.** *For a tensor $\boldsymbol{\mathcal{T}} \in \mathbb{R}^{I \times J \times K}$ with $n = \max(I, J, K)$ and any factor matrices on the first two dimension as $\mathbf{A} \in \mathbb{R}^{I \times R}$ and $\mathbf{B} \in \mathbb{R}^{J \times R}$. If a step of ALS on the third dimension gives $\mathbf{C}_{opt}$, then a step of SPALS that samples $m = \Theta(R^2 \log n / \epsilon^2)$ rows produces $\overline{\mathbf{C}}$ satisfying*

$$\left\| \boldsymbol{\mathcal{T}} - [\![\mathbf{A}, \mathbf{B}, \overline{\mathbf{C}}]\!] \right\|^2 < \left\| \boldsymbol{\mathcal{T}} - [\![\mathbf{A}, \mathbf{B}, \mathbf{C}_{opt}]\!] \right\|^2 + \epsilon \|\boldsymbol{\mathcal{T}}\|^2.$$

*Proof.* Denote the sample-and-rescale matrix as $\mathbf{S} \in \mathbb{R}^{m \times IJ}$. By Corollary E.3, we have that $\left\| \mathbf{T}_{(3)} (\mathbf{B} \odot \mathbf{A}) - \mathbf{T}_{(3)} \mathbf{S}^{\top} \mathbf{S} (\mathbf{B} \odot \mathbf{A}) \right\| \leq \epsilon \|\boldsymbol{\mathcal{T}}\|$. Together with Lemma E.1, we can conclude. $\square$

Note that the approximation error of our algorithm does not accumulate over iterations. Similar to the stochastic gradient descent algorithm, the error occurred in the previous iterations can be addressed in the subsequent iterations.

## 4.3 Extensions on Other Tensor Related Applications

**Importance Sampling SGD on CP Decompostion** We can incorporate importance sampling in the stochastic gradient descent algorithm for CP decomposition. The gradient follows the form

$$\frac{\partial}{\partial \mathbf{C}} \|\boldsymbol{\mathcal{T}} - [\![\mathbf{A}, \mathbf{B}, \mathbf{C}]\!]\|^2 = \mathbf{T}_{(3)} (\mathbf{B} \odot \mathbf{A}).$$

By sampling rows according to proposed distribution, it reduces the per-step variance via importance sampling [26]. Our result addresses the computational difficulty of finding the appropriate importance.

**Sampling ALS on Higher-Order Singular Value Decomposition (HOSVD)** For solving the HOSVD [13] on tensor, the Kronecker product is involved instead of the Khatri-Rao product. In Appendix D, we prove similar leverage score approximation results for Kronecker product. In fact, for Kronecker product, our "approximation" provides the exact leverage score.

**Theorem 4.2.** *For matrix* $\mathbf{A} \in \mathbb{R}^{I \times M}$ *and matrix* $\mathbf{B} \in \mathbb{R}^{J \times N}$, *where* $I > M$ *and* $J > N$, *let* $\tau_i^{\mathbf{A}}$ *and* $\tau_j^{\mathbf{B}}$ *be the statistical leverage score of the* $i$-*th and* $j$-*th row of* $\mathbf{A}$ *and* $\mathbf{B}$, *respectively. Then, for matrix* $\mathbf{A} \otimes \mathbf{B} \in \mathbb{R}^{IJ \times MN}$ *with statistical leverage score* $\tau_{i,j}^{\mathbf{A} \otimes \mathbf{B}}$ *for the* $(iJ + j)$-*th row, we have* $\tau_{i,j}^{\mathbf{A} \otimes \mathbf{B}} = \tau_i^{\mathbf{A}} \tau_j^{\mathbf{B}}$.

# 5 Related Works

CP decomposition is one of the simplest, most easily-interpretable tensor decomposition. Fitting it in an ALS fashion is still considered as the state-of-art in the recent tensor analytics literature [37]. The most widely used implementation of ALS is the MATLAB Tensor Toolbox [21]. It directly performs the analytic solution of ALS steps. There is a line of work on speeding up this procedure in distributed/parallel/MapReduce settings [20, 19, 5, 33]. Such approaches are compatible with our approach, as we directly reduce the number of steps by sampling. A similar connection holds for works achieving more efficient computation of KRP steps of the ALS algorithm such as in [32].

The applicability of randomized numerical linear algebra tools to tensors was studied during their development [28]. Within the context of sampling-based tensor decomposition, early work has been published in [36, 35] that focuses though on Tucker decomposition. In [30], sampling is used as a means of extracting small representative sub-tensors out of the initial input, which are further decomposed via the standard ALS and carefully merged to form the output. Another work based on an a-priori sampling of the input tensor can be found in [2]. However, recent developments in randomized numerical linear algebra often focused on over-constrained regression problems or low rank matrices. The incorporation of such tools into tensor analytics routines was fairly recent [31, 37]

Most closely related to our algorithm are the routines from [37], which gave a sketch-based CP decomposition inspired by the earlier work in [31]. Both approaches only need to examine the factorization at each iteration, followed by a number of updates that only depends on rank. A main difference is that the sketches in [37] moves the non-zeroes, while our sampling approach removes many entries instead. Their algorithm also performs a subsequent FFT step, while our routine always works on subsets of the matricizations. Our method is much more suitable for sparse tensors. Also, our routine can be considered as data dependent randomization, which enjoys better approximation accuracy than [37] in the worst case.

For direct comparison, the method in [37] and ours both require $\text{nnz}(\mathcal{T})$ preprocessing at the beginning. Then, for each iteration, our method requires $\tilde{O}(nr^3)$ operations comparing with $O(r(n + Bb \log b) + r^3)$ for [37]. Here $B$ and $b$ for [37] are parameters for the sketching and need to be tuned for various applications. Depending on the target accuracy, $b$ can be as large as the input size: on the cube synthetic tensors with $n = 10^3$ that the experiments in [37] focused on, $b$ was set to between $2^{14} \approx \times 10^3$ and $2^{16} \approx 6 \times 10^4$ in order to converge to good relative errors.

From a distance, our method can be viewed as incorporating randomization into the intermediate steps of algorithms, and can be viewed as higher dimensional analogs of weighted SGD algorithms [39]. Compared to more global uses of randomization [38], these more piecemeal invocations have several advantages. For high dimensional tensors, sketching methods need to preserve all dimensions, while the intermediate problems only involve matrices, and can often be reduced to smaller dimensions. For approximating a rank $R$ tensor in $d$ dimensions to error $\epsilon$, this represents the difference between $poly(R, \epsilon)$ and $\frac{R}{\epsilon}^d$. Furthermore, the lower cost of each step of alternate minimization makes it much easier to increase accuracy at the last few steps, leading to algorithms that behave the same way in the limit. The wealth of works on reducing sizes of matrices while preserving objectives such as $\ell_p$ norms, hinge losses, and M-estimators [11, 10, 8, 7] also suggest that this approach can be directly adapted to much wider ranges of settings and objectives.

# 6 Experimental Results

We implemented and evaluated our algorithms in a single machine setting. The source code is available online[1]. Experiments are tested on a single machine with two Intel Xeon E5-2630 v3 CPU and 256GB memory. All methods are implemented in C++ with OpenMP parallelization. We report averages from 5 trials.

**Dense Synthetic Tensors** We start by comparing our method against the sketching based algorithm from [37] in the single thread setting as in their evaluation. The synthetic data we tested are third-order tensors with dimension $n = 1000$, as described in [37]. We generated a rank-1000 tensor with harmonically decreasing weights on rank-1 components. And then after normalization, random Gaussian noise with noise-to-signal $nsr = 0.1, 1, 10$ was added. As with previous experimental evaluations [37], we set target rank to $r = 10$. The performances are given in Table 1a. We vary the sampling rate of our algorithm, i.e., SPALS($\alpha$) will sample $\alpha r^2 \log^2 n$ rows at each iteration.

|  | $nsr = 0.1$ | | $nsr = 1$ | | $nsr = 10$ | |
|---|---|---|---|---|---|---|
|  | error | time | error | time | error | time |
| ALS-dense | 0.27 | 64.8 | 1.08 | 66.2 | 10.08 | 67.6 |
| sketch(20, 14) | 0.45 | 6.50 | 1.37 | 4.70 | 11.11 | 4.90 |
| sketch(40, 16) | 0.30 | 16.0 | 1.13 | 12.7 | 10.27 | 12.4 |
| ALS-sparse | 0.24 | 501 | 1.09 | 512 | 10.15 | 498 |
| SPALS(0.3) | 0.20 | 1.76 | 1.14 | 1.93 | 10.40 | 1.92 |
| SPALS(1) | 0.18 | 5.79 | 1.10 | 5.64 | 10.21 | 5.94 |
| SPALS(3.0) | 0.21 | 15.9 | 1.09 | 16.1 | 10.15 | 16.16 |

(a) Running times per iterations in seconds and errors of various alternating least squares implementations

|  | error | time |
|---|---|---|
| ALS-sparse | 0.981 | 142 |
| SPALS(0.3) | 0.987 | 6.97 |
| SPALS(1) | 0.983 | 15.7 |
| SPALS(3.0) | 0.982 | 38.9 |

(b) Relative error and running times per iteration on the Amazon review tensor with dimensions $2.44e6 \times 6.64e6 \times 9.26e4$ and 2.02 billion non-zeros

On these instances, a call to SPALS with rate $\alpha$ samples was about $4.77\alpha \times 10^3$ rows, and as the tensor is dense, $4.77\alpha \times 10^6$ entries. The correspondence between running times and rates demonstrate the sublinear runtimes of SPALS with low sampling rates. Comparing with the [37], our algorithm employs data dependent random sketch with minimal overhead, which yields significantly better precision with similar amount of computation.

**Sparse Data Tensor** Our original motivation for SPALS was to handle large sparse data tensors. We ran our algorithm on a large-scale tensor generated from Amazon review data [24]. Its sizes and convergences of SPALS with various parameters are in Table 1b. We conduct the experiments in parallel with 16 threads. The Amazon data tensor has a much higher noise to signal ratio than our other experiments which common for large-scale data tensors: Running deterministic ALS with rank 10 on it leads to a relative error of $98.1\%$. SPALS converges rapidly towards a good approximation with only a small fraction of time comparing with the ALS algorithm.

# 7 Discussion

Our experiments show that SPALS provides notable speedup over previous CP decomposition routines on both dense and sparse data. There are two main sources of speedups: (1) the low target rank and moderate individual dimensions enable us to compute leverage scores efficiently; and (2) the simple representations of the sampled form also allows us to use mostly code from existing ALS routines with minimal computational overhead. It is worth noting that in the dense case, the total number of entries accessed during all 20 iterations is far fewer than the size of $\mathcal{T}$. Nonetheless, the adaptive nature of the sampling scheme means all the information from $\mathcal{T}$ are taken into account while generating the first and subsequent iterations. From a randomized algorithms perspective, the sub-linear time sampling steps bear strong resemblances with stochastic optimization routines [34]. We believe more systematically investigating such connections can lead to more direct connections between tensors and randomized numerical linear algebra, and in turn further algorithmic improvements.

**Acknowledgments**
This work is supported in part by the U. S. Army Research Office under grant number W911NF-15-1-0491, NSF Research Grant IIS-1254206 and IIS-1134990. The views and conclusions are those of the authors and should not be interpreted as representing the official policies of the funding agency, or the U.S. Government.

## Footnotes

[1] https://github.com/dehuacheng/SpAls

# References

[1] B. Barak, J. A. Kelner, and D. Steurer. Dictionary learning and tensor decomposition via the sum-of-squares method. In *STOC*, 2015.

[2] S. Bhojanapalli and S. Sanghavi. A New Sampling Technique for Tensors. *ArXiv e-prints*, 2015.

[3] J. D. Carroll and J.-J. Chang. Analysis of individual differences in multidimensional scaling via an n-way generalization of "eckart-young" decomposition. *Psychometrika*, 1970.

[4] D. Cheng, Y. Cheng, Y. Liu, R. Peng, and S.-H. Teng. Spectral sparsification of random-walk matrix polynomials. *arXiv preprint arXiv:1502.03496*, 2015.

[5] J. H. Choi and S. Vishwanathan. Dfacto: Distributed factorization of tensors. In *NIPS*, 2014.

[6] K. L. Clarkson and D. P. Woodruff. Low rank approximation and regression in input sparsity time. In *STOC*, 2013.

[7] K. L. Clarkson and D. P. Woodruff. Input sparsity and hardness for robust subspace approximation. In *FOCS*, 2015.

[8] K. L. Clarkson and D. P. Woodruff. Sketching for m-estimators: A unified approach to robust regression. In *SODA*, 2015.

[9] M. B. Cohen, Y. T. Lee, C. Musco, C. Musco, R. Peng, and A. Sidford. Uniform sampling for matrix approximation. In *ITCS*, 2015.

[10] M. B. Cohen and R. Peng. $\ell_p$ row sampling by Lewis weights. In *STOC*, 2015.

[11] A. Dasgupta, P. Drineas, B. Harb, R. Kumar, and M. W. Mahoney. Sampling algorithms and coresets for \ell_p regression. *SIAM Journal on Computing*, 2009.

[12] L. De Lathauwer and B. De Moor. From matrix to tensor: Multilinear algebra and signal processing. In *Institute of Mathematics and Its Applications Conference Series*, 1998.

[13] L. De Lathauwer, B. De Moor, and J. Vandewalle. A multilinear singular value decomposition. *SIAM journal on Matrix Analysis and Applications*, 2000.

[14] V. De Silva and L.-H. Lim. Tensor rank and the ill-posedness of the best low-rank approximation problem. *SIAM J. Matrix Anal. Appl.*, 2008.

[15] P. Drineas, M. W. Mahoney, S. Muthukrishnan, and T. Sarlós. Faster least squares approximation. *Numerische Mathematik*, 2011.

[16] R. Ge, F. Huang, C. Jin, and Y. Yuan. Escaping from saddle points - online stochastic gradient for tensor decomposition. In *COLT*, 2015.

[17] R. A. Harshman. Foundations of the parafac procedure: Models and conditions for an" explanatory" multi-modal factor analysis. 1970.

[18] C. J. Hillar and L.-H. Lim. Most tensor problems are np-hard. *Journal of the ACM (JACM)*, 2013.

[19] I. Jeon, E. E. Papalexakis, U. Kang, and C. Faloutsos. Haten2: Billion-scale tensor decompositions. In *ICDE*, 2015.

[20] U. Kang, E. Papalexakis, A. Harpale, and C. Faloutsos. Gigatensor: scaling tensor analysis up by 100 times-algorithms and discoveries. In *KDD*, 2012.

[21] T. G. Kolda and B. W. Bader. Tensor decompositions and applications. *SIAM review*, 2009.

[22] M. Li, G. Miller, and R. Peng. Iterative row sampling. In *FOCS*, 2013.

[23] M. W. Mahoney. Randomized algorithms for matrices and data. *Foundations and Trends® in Machine Learning*, 2011.

[24] J. McAuley and J. Leskovec. Hidden factors and hidden topics: understanding rating dimensions with review text. In *RecSys*, 2013.

[25] X. Meng and M. W. Mahoney. Low-distortion subspace embeddings in input-sparsity time and applications to robust linear regression. In *STOC*, 2013.

[26] D. Needell, R. Ward, and N. Srebro. Stochastic gradient descent, weighted sampling, and the randomized kaczmarz algorithm. In *NIPS*, 2014.

[27] J. Nelson and H. L. Nguyên. Osnap: Faster numerical linear algebra algorithms via sparser subspace embeddings. In *FOCS*, 2013.

[28] N. H. Nguyen, P. Drineas, and T. D. Tran. Tensor sparsification via a bound on the spectral norm of random tensors. *CoRR*, 2010.

[29] A. Novikov, D. Podoprikhin, A. Osokin, and D. P. Vetrov. Tensorizing neural networks. In *NIPS*, 2015.

[30] E. E. Papalexakis, C. Faloutsos, and N. D. Sidiropoulos. Parcube: Sparse parallelizable tensor decompositions. In *Machine Learning and Knowledge Discovery in Databases*. Springer, 2012.

[31] N. Pham and R. Pagh. Fast and scalable polynomial kernels via explicit feature maps. In *KDD*, 2013.

[32] A.-H. Phan, P. Tichavsky, and A. Cichocki. Fast alternating ls algorithms for high order candecomp/parafac tensor factorizations. *Signal Processing, IEEE Transactions on*, 2013.

[33] S. Smith, N. Ravindran, N. D. Sidiropoulos, and G. Karypis. Splatt: Efficient and parallel sparse tensor-matrix multiplication. *29th IEEE International Parallel & Distributed Processing Symposium*, 2015.

[34] T. Strohmer and R. Vershynin. A randomized kaczmarz algorithm with exponential convergence. *JFAA*, 2009.

[35] J. Sun, S. Papadimitriou, C.-Y. Lin, N. Cao, S. Liu, and W. Qian. Multivis: Content-based social network exploration through multi-way visual analysis. In *SDM*. SIAM, 2009.

[36] C. E. Tsourakakis. Mach: Fast randomized tensor decompositions. In *SDM*. SIAM, 2010.

[37] Y. Wang, H.-Y. Tung, A. J. Smola, and A. Anandkumar. Fast and guaranteed tensor decomposition via sketching. In *NIPS*, 2015.

[38] D. P. Woodruff. Sketching as a tool for numerical linear algebra. *Foundations and Trends® in Theoretical Computer Science*, 2014.

[39] J. Yang, Y. Chow, C. Ré, and M. W. Mahoney. Weighted sgd for $\ell_p$ regression with randomized preconditioning. In *SODA*, 2016.

[40] R. Yu, D. Cheng, and Y. Liu. Accelerated online low rank tensor learning for multivariate spatiotemporal streams. In *ICML*, pages 238–247, 2015.

